# A Motion-aware Spatio-temporal Graph for Video Salient Object Ranking

Hao Chen[1,2], Yufei Zhu[1,2], and Yongjian Deng[3]

[1]School of Computer Science and Engineering, Southeast University, Nanjing, China
[2]Key Laboratory of New Generation Artificial Intelligence Technology and Its Interdisciplinary Applications (Southeast University), Ministry of Education, China.
[3]College of Computer Science, Beijing University of Technology, Beijing, China
{haochen303, 220232213}@seu.edu.cn, yjdeng@bjut.edu.cn

## Abstract

Video salient object ranking aims to simulate the human attention mechanism by dynamically prioritizing the visual attraction of objects in a scene over time. Despite its numerous practical applications, this area remains underexplored. In this work, we propose a graph model for video salient object ranking. This graph simultaneously explores multi-scale spatial contrasts and intra-/inter-instance temporal correlations across frames to extract diverse spatio-temporal saliency cues. It has two advantages: 1. Unlike previous methods that only perform global inter-frame contrast or compare all proposals across frames globally, we explicitly model the motion of each instance by comparing its features with those in the same spatial region in adjacent frames, thus obtaining more accurate motion saliency cues. 2. We synchronize the spatio-temporal saliency cues in a single graph for joint optimization, which exhibits better dynamics compared to the previous stage-wise methods that prioritize spatial cues followed by temporal cues. Additionally, we propose a simple yet effective video retargeting method based on video saliency ranking. Extensive experiments demonstrate the superiority of our model in video salient object ranking and the effectiveness of the video retargeting method. Our codes/models are released at https://github.com/zyf-815/VSOR/tree/main.

## 1 Introduction

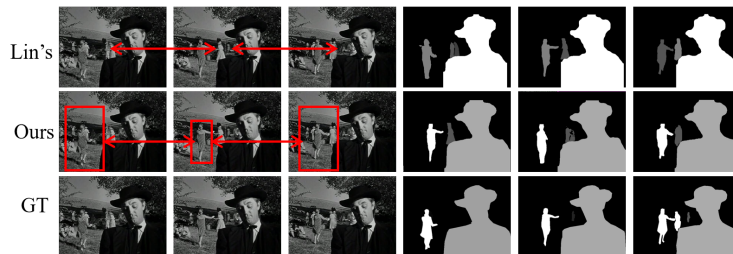

Figure 1: Previous methods (*e.g.*, [1]) in video SOR primarily focus on capturing temporal correlations between global features across frames, resulting in limited ability to effectively model temporal saliency. These methods tend to emphasize objects with prominent static saliency cues (*i.e.*, the man right), whereas our proposed model places greater emphasis on instance-wise temporal correlation, resulting in more accurate results (*i.e.*, the walking woman left).

Salient Object Ranking (SOR) [2–6] is to mimic the visual priority coding mechanism [7] in the human visual system to rank the attractiveness of objects in a scene. It has a large range of applications such as image caption [8, 9], image compression [10, 11] and image retargeting [12, 13]. Previous research primarily focuses on ranking salient objects in static images. However, Video Salient Object Ranking (VSOR) [1, 14] holds specific potential for applications such as video summarization and surveillance. It also presents distinct challenges due to the changing saliency ranking of objects within a sequence. Nonetheless, this area remains understudied.

SVSNet [14] is a pioneering work in Video Salient Object Ranking (VSOR) that leverages the proportion of eye fixation points on each object to infer the relative saliency ranking among objects. The model employs frame-level LSTM to capture global temporal information for saliency detection and fixation prediction. However, this approach depends on the availability of eye fixation labels, which are often absent. Moreover, it relies on an external object detector [15] to generate bounding boxes for computing fixation density, rendering the network non end-to-end. Additionally, it lacks instance-wise temporal modeling.

Recently, Lin [1] propose an end-to-end solution by integrating the object detector into the VSOR model for joint training. The VSOR results are obtained by initially modeling the correlations among all objects in each frame to perform individual image SOR. Subsequently, these static image SOR results are combined by considering inter-frame correlations to produce the final outcome.

Although this approach eliminates the dependency on eye fixations and achieves end-to-end video SOR, it still has a significant issue: as shown in the $1^{st}$ row in Figure 1, it directly builds correlations among all instances across frames without explicitly differentiating instance-wise motion, thus remaining agnostic to instance-level motion saliency.

As the human visual system naturally focuses more on moving objects, our core motivation is to explicitly explore the inter-frame variation for each instance. Considering the lack of annotations for object tracking, we propose a simple and effective method to capture instance-level motion information. as shown in the $2^{nd}$ row in Figure 1, since the absolute positions of objects between two frames do not change drastically, we can approximate the instance motion by comparing the features of an instance at the same position across frames. Additionally, we double the area of the bounding boxes for comparison to reduce errors caused by camera motion.

Spatial saliency in a frame is also indispensable for VSOR. Drawing inspiration from [2], we also conduct multi-scale contrast—between an instance and its larger local region, the global image, and another instance—in order to explore diverse spatial saliency cues in a graph neural network (GNN).

The following question is how to combine and optimize the temporal and spatial graphs. In previous work [1], a sequential combination of spatial and temporal graphs is adopted, but in our opinion, this approach lacks adaptability and may exhibit bias towards one side. For instance, when initially aggregating spatial information, the instance features contain spatial saliency signals, which can hinder the extraction of temporal features. Therefore, we propose to jointly optimize the various spatial and temporal relations within a unified graph. This joint graph facilitates more comprehensive and dynamic interactions among instances and frames, consequently enhancing the fusion sufficiency of diverse saliency cues and improving the ranking results.

As an emerging field, we also explore its application scenarios. The ranking of salient objects in videos reveals the varying degrees of attention and trajectory shifts towards different regions during video playback. This information happens to be crucial for the task of video retargeting [13, 16–18, 12]. Therefore, we propose a method that applies video saliency object ranking to video retargeting. Our method is straightforward. We assign different weights to salient objects based on their saliency rankings, and then locate the saliency centroid of the current frame as the center for retargeting. Experimental results demonstrate that our proposed simple method effectively preserves the regions of interest with greater accuracy and comprehensiveness while mitigating background interference. Additionally, it exhibits better inter-frame continuity.

In summary, our contributions are as follows:

(1) Introducing a novel unified spatial-temporal graph that explicitly integrates motion-aware temporal correlation and adaptively integrate spatial and temporal saliency cues.

(2) Proposing a simple yet efficient video retargeting method based on VSOR.

(3) Conducting experiments that validate the efficacy of our temporal saliency cues, the strategy for fusing spatial and temporal cues, and the superior performance of our VSOR and retargeting models compared to state-of-the-art methods.

## 2 Related Work

### 2.1 Salient Object Ranking on Images

Islam [3] first notice that it is difficult for people to form a unified judgment about which object is the most salient in a scene. Therefore, they propose the problem of relative saliency, namely saliency ranking, and use a stage-wise refinement model to gradually detect objects with different saliency levels. However, the use of pixel-level saliency ranking often results in varying saliency values for different pixels in the same instance.

To address this problem, a substantial amount of subsequent work [4, 5, 2, 6] has proposed instance-level saliency ranking tasks. Siris [4] infer the saliency ranking by modeling the human attention transfer mechanism with a combined bottom-up and top-down attention framework. Considering the influence of low-level features such as the position and size of objects on instance saliency, Fang [5] proposed a PPA module to model position information. Liu [2] utilize graph neural networks to comprehensively consider the spatial correlations, including the contrast between instances, the distinctiveness of a instance in its local/global contexts, as well as prior knowledge of categories.

These methods achieve good results in saliency ranking on static images, but for videos, the motion of objects greatly affects their saliency order. Therefore, existing saliency ranking models for images cannot achieve good results and specific designs towards the inter-frame relations are required.

### 2.2 Video Salient Object Detection

Another topic related to our task is video salient object detection. Unlike static image salient object detection, it [19–21] needs to jointly consider the different saliency cues generated by the motion information across frames and static appearance. Li [19] use optical flow to model temporal features and combine spatial-temporal relationships through attention mechanisms. Fan [20] further highlight the attention shift problem resulted from motion and propose a recurrent model to model temporal dynamics and attention-shift jointly. Liu [21] consider single-frame appearance information, long/short-term motion and spatial–temporal cues jointly in a transformer.

However, these works focus on detecting the most salient object from a video, while salient object ranking task is more complicated and with more temporal dynamics. Also, these works fails to explore the instance-wise motion cues, which is crucial to change saliency ranking results.

### 2.3 Video Salient Object Ranking

Wang [14] first introduce the video salient object ranking task and propose a multi-task network SVSNet. However, SVSNet only utilizes LSTM to capture global temporal features, ignoring the instance-level features that are indispensable to saliency ranking. Furthermore, the utilization of extra detectors in SVSNet leads to the model not being end-to-end. Lin [1] address these problems by proposing an end-to-end architecture, where the relationships between objects and temporal information are combined to assign rankings to each object. However, at the temporal level, Lin only use global features as saliency cues, ignoring the motion information of each instance.

Instead, we optimize the multi-scale spatial contrasts, inter-frame instance relations, and instance-wise local motion at the same time, and the resulting network is equipped with comprehensive saliency cues and enjoys better fusion adaptivity.

## 3 Method

### 3.1 Overview

As shown in Figure 2, our proposed VSOR framework firstly segments instances based on Mask R-CNN [22], and then enhance instance features through attention mechanisms [23] and introduce

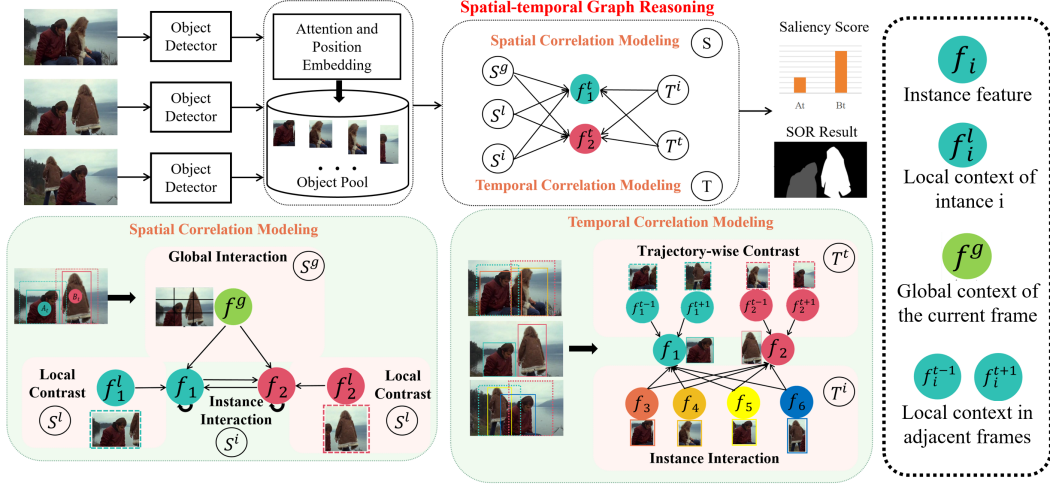

Figure 2: The main architecture of our model. Firstly, we obtain instances of each frame through an object detector and extract features of each instance using an attention mechanism and position embedding. Then, we utilize a spatial-temporal graph reasoning network to fuse spatial-temporal saliency cues, ultimately obtaining saliency scores and final results.

position and scale saliency prior using PPA [5]. Subsequently, A spatial-temporal graph neural network is crafted to jointly optimize multi-scale spatial and temporal contrast cues to generate integrated instance-level saliency features. Finally, a fully connected neural network is employed to infer saliency scores, which, combined with the results of instance segmentation, yield the final saliency ranking map. This model can be trained end-to-end.

## 3.2 Unified Spatial-temporal Graph for VSOR

After obtaining the detection results through Mask R-CNN, instance features $f_i$ are introduced into the Spatial-temporal Graph Reasoning model. Our graph reasoning model includes two views: spatial correlation modeling and temporal correlation modeling. In spatial correlation modeling, we draw inspiration from [2] to introduce the interaction between instances $S^i$, the comparison between instances and their local contextual features $S^l$, as well as the comparison between instances and global contextual features $S^g$. We denote local contextual features as $f_i^l$, where $i$ means the instance $i$, and global features are represented as $f^g$. In temporal correlation module, we introduce the motion-aware contrast $T^t$ and instance interaction $T^i$. $f_i^{\{t-1,t+1\}}$ means the motion features in frame $t-1$ and $t+1$ of instance $i$.

### 3.2.1 Spatial Correlation Modeling

**Interaction between instances.** The competition among multiple instances for saliency highlights the importance of inter-instance interaction in saliency ranking. Therefore, we establish edges between instances to capture their correlations and the aggregation function is represented as:

$$h_{N_i^r} = \sum_{j=1}^{N} \alpha_{ij}^r W_a^r f_j; \qquad \alpha_{ij}^r = \frac{1}{N} ReLU((W_\alpha^r)^T (U^r f_i || V^r f_j)) \tag{1}$$

where $N$ means the number of instances and $\alpha_{ij}^r$ is the attention weight modulating the edge between instance $i$ and $j$, and $W_a^r$, $U^r$ and $V^r$ are projection matrixes. $||$ means connection between features.

**Local contrast and global interaction.** It is clear that the saliency of an instance is influenced by both its local and global contexts [2]. To capture the local contextual feature $f_i^l$, we expand the bounding box of instance $i$ by doubling its size. For the global contextual features, we divide the image into $3 \times 3$ and utilize average pooling to extract each feature.

### 3.2.2 Temporal Correlation Modeling

**Instance interaction.** Similar to the interaction between instances in the spatial correlation module, the interaction between instances in different frames can also have an impact on the saliency of instances. Hence, we begin temporal correlation modeling by establishing edges between instances in adjacent frames and instances in the current frame. And the aggregation function is represented as:

$$h_{N_i^{t_i}} = \sum_{j=1}^{N_{t-1}+N_{t+1}} \alpha_{ij}^i W_a^i f_j; \qquad \alpha_{ij}^i = \frac{1}{N_{t-1}+N_{t+1}} ReLU((W_\alpha^i)^T (U^i f_i \| V^i f_j)) \quad (2)$$

where $N_{t-1} + N_{t+1}$ means the total number of instances in frame $t-1$ and $t+1$ and $\alpha_{ij}^i$ is the attention weight modulating the edge between instance $i$ and $j$.

**Motion-wise contrast.** Merely constructing temporal connections by modeling instance interaction is not enough to model temporal saliency, as the human visual system tends to prioritize moving objects. Hence, it is necessary to introduce independent construction of motion representations for each instance. To achieve this, for instance $i$, we double its bounding box and apply ROIAlign to extract features within this region in the previous and next frames and represent them as $f_i^{t-1}$ and $f_i^{t+1}$. By comparing the instance features with its local contextual features $f_i^{t-1}$ and $f_i^{t+1}$ in adjacent frames, we are able to model the saliency from the motion trajectory for instance $i$. The aggregation function is represented as:

$$h_{N_i^{tt}} = \sum_{j=t-1,t+1} \alpha_{ij}^t W_a^t f_i^j; \qquad \alpha_{ij}^t = \frac{1}{2} ReLU((W_\alpha^t)^T (U^t f_i \| V^t f_j)) \quad (3)$$

### 3.2.3 Spatial-temporal Fusion

Another important issue is how to integrate spatial-temporal cues. An intuitive idea is to aggregate spatial and temporal correlation modeling results in a serial manner as done in [1]. However, this sequential integration strategy may embed spatial clues in the instance features, which is counter-productive to extracting temporal saliency. To solve this problem, we propose to unify spatial and temporal correlation modeling and optimize them jointly.

Drawing inspiration from the graph updating methods outlined in [2], we extend the overall graph to $K$ parallel subgraphs to stabilize the learning process and enrich the learned node interaction connections. Each subgraph independently learns node interactions. As a result, the instance feature can be represented as:

$$f_i^{\tilde{t}} = f_i^t + \Big\|_{k=1}^{K} (W_u^{r,k} h_{N_i^r}^k + W_u^{l,k} h_{N_i^l}^k + W_u^{g,k} h_{N_i^g}^k + W_u^{t_i,k} h_{N_i^{t_i}}^k + W_u^{t_t,k} h_{N_i^{tt}}^k) \quad (4)$$

### 3.3 Loss Function

Our loss function can be computed as:

$$L = L_{det} + L_{SOR} \quad (5)$$

$L_{det} = L_{cls} + L_{box} + L_{mask}$ is defined in Mask R-CNN [22] to be used to detect the salient instance, where $L_{cls}$, $L_{box}$, $L_{mask}$ denote the classification loss, regression loss and mask loss respectively. $L_{SOR}$ is the ranking loss proposed by [2]:

$$L_{SOR} = \sum_{q=1}^{C_N^2} \beta_q log(1 + exp((-s_{q1} + s_{q2}))) \quad (6)$$

where $\beta_q$ is the dynamic loss weight and $s_{q1}, s_{q2}$ are saliency scores predicted by our network.

## 4 Experiment

### 4.1 Experimental Setup

**Dataset collection and annotation.** Wang [14] propose a video saliency ranking task and the first dataset RVSOD, but they only focus on frame-level saliency and ignore the instance-level saliency. Therefore, RVSOD lacks instance-level annotations to evaulate VSOR models. To address this limitation, we utilize the manually annotated masks provided in RVSOD to obtain instance masks and assign the saliency ranking score to each instance based on the distribution of fixation points. In this way, the instance-level annotations are generated.

However, the RVSOD dataset suffers from two notable limitations: (a) a lack of complex scenes, and (b) most scenes consist of only one salient instance, rendering it unsuitable for evaluating the ability of salient object ranking (SOR). To address these issues, we utilize the video saliency detection dataset DAVSOD [20] to extract saliency ranking results, thereby making it more appropriate for video saliency ranking tasks. In order to fully utilize the DAVSOD, we classify the scenes in DAVSOD into three categories: (a) all video frames contain only one salient instance, (b) some video frames contain one salient instance, and (c) all video frames contain multiple salient instances. We discard the scenario described in (a) and proceed to divide the remaining scenes into training and testing sets in a 4:1 ratio for both scenarios (b) and (c). The generation of the instance-level annotations is the same as the RVSOD. The process of generating salient ranks and the statistical analysis of DAVSOD are presented in the supplementary material in section A.1.

**Evaluation metrics.** SA-SOR [2] and Mean Absolute Error (MAE) are used to evaluate the performance of our model in ranking and segmentation. SA-SOR can reflect both segmentation and ranking performance at the same time which make it appear more reliable than the original SOR [3] or the SSOR [4] metric. MAE serves as a reliable indicator of segmentation quality.

**Implementation details.** We adopt the training strategy outlined in [3], utilizing the pre-trained Mask R-CNN learned on the MS-COCO 2017 training split. The settings remain consistent throughout the experiment, with the box head solely dedicated to classifying objects and non-objects. Stochastic gradient descent (SGD) is employed to optimize the loss function. To facilitate training, we implement a warm-up strategy, commencing with an initial learning rate of 5e-3. At the 420,000 and 500,000 steps, the learning rate is reduced by a factor of 10.

Once the training of our detection branch is complete, we proceed to incorporate our graph module and fine-tune the entire network using two separate datasets - RVSOD and DAVSOD. Throughout this process, we set the batch size to 1 and the maximum iteration count to 200,000. For optimization, we utilize the Adam optimizer with an initial learning rate of 5e-6. At the 80,000th and 150,000th steps, the learning rate is reduced by a factor of 10. Our model is implemented using PyTorch and all experiments are conducted on a NVIDIA RTX4090.

### 4.2 Compared to State-of-the-art Methods

Table 1: Quantitative results on the generated datasets. The bigger the SA-SOR, the better the performance of ranking. The smaller the MAE, the better the performance of segmentation. The best results are indicated in bold. The method by Lin [1] is not yet open-source, so we only asked them for the results on RVSOD.

| Method | | RVSOD | | DAVSOD | |
|---|---|---|---|---|---|
| | | SA-SOR | MAE | SA-SOR | MAE |
| SOR | Fang [5] | 0.350 | 0.0984 | 0.157 | 0.0760 |
| | Liu [2] | 0.563 | 0.0728 | 0.179 | 0.0639 |
| | PSR [24] | 0.405 | 0.074 | - | - |
| | SeqRank [25] | 0.512 | 0.0761 | - | - |
| VSOD | DCFNet [26] | - | 0.1180 | - | 0.0684 |
| | SCOTCH [27] | - | 0.1230 | - | 0.1126 |
| VSOR | Lin [1] | 0.560 | 0.0745 | - | - |
| | Ours | **0.603** | **0.0698** | **0.207** | **0.0626** |

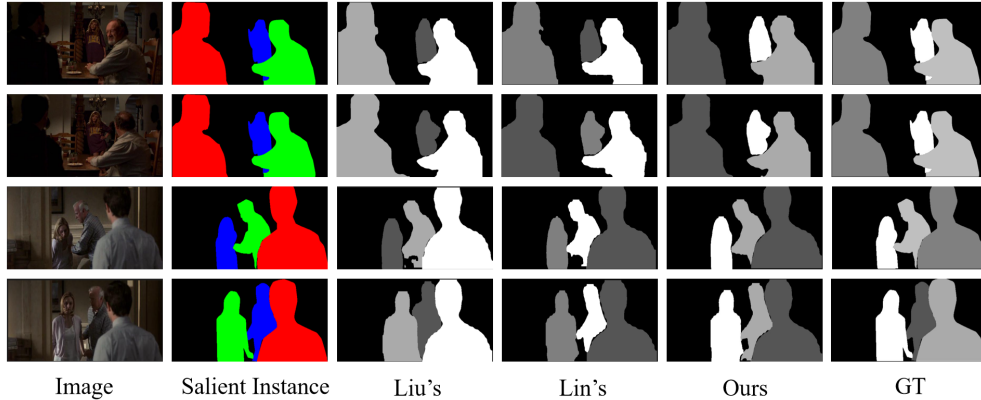

| Image | Salient Instance | Liu's | Lin's | Ours | GT |

Figure 3: Visual results with our model and other sota methods. The scenarios are chosen from $RVSOD/actioncliptest00256$ and $RVSOD/actioncliptest00707$ respectively.

**Quantitative results.** We compare our proposed network with six other state-of-the-art salient object ranking methods, including four image-based SOR methods, two video salient object detection methods and one available video-based SOR method. As shown in Table 1, our model largely outperforms previous methods on both SA-SOR and MAE, indicating the advantages of our model in modeling and fusing spatial-temporal saliency. We discuss the reasons for poor results on DAVSOD in the supplementary material in section A.2.

**Visual results.** We also perform visual comparisons to verify the advantages of our method. As shown in Figure 3, the results of [2] highlight those objects with distinguished static saliency cues (e.g., the green and red instances) such as larger size, closer distance or clearer appearance, while the blue instance with less static saliency cues yet rich motion cues is ranked the least salient, as this method only considers multi-scale spatial contrast and instance relation in an individual image. Although the method of [1] additionally takes the temporal cues into account, its rankings are also biased towards static saliency cues as it simply models temporal relations across the global representation of each frame, ignoring the instance-wise correspondence moving across frames.

In contrast, our method explicitly considers the motion cues of the instance and perform instance-level spatial and temporal reasoning in a unified graph. Our method successfully distinguish instances with less static saliency cues but semantically rich motion cues (e.g., the blue instances should be the central character).

## 4.3 Ablation Study

Table 2: Ablation study for our temporal interaction. Our baseline is the Liu's [2] model without person prior. Then we consider three temporal features: global-aware temporal information(GTRM), instance-aware features(ITRM), and motion-aware features(MTRM). Our final method combines the above three temporal features simultaneously.

| Method | SA-SOR | |
|---|---|---|
| | RVSOD | DAVSOD |
| Baseline( w/o TRM ) | 0.563 | 0.179 |
| Baseline + GTRM | 0.585 | 0.186 |
| Baseline + ITRM | 0.591 | 0.194 |
| Baseline + MTRM | 0.587 | 0.195 |
| Ours(joint) | **0.603** | **0.207** |

**Effectiveness on the temporal interaction.** We firstly study the advantages of our motion-aware temporal correlation modeling. Specifically, the model that excludes temporary relationship modeling serves as the baseline. Then, we have explored three types of temporary relationship modeling, including GTRM (global-aware temporal relation modeling), ITRM (instance-aware temporal relation modeling) and our motion-aware temporal relation modeling (MTRM). For GTRM, we utilize the

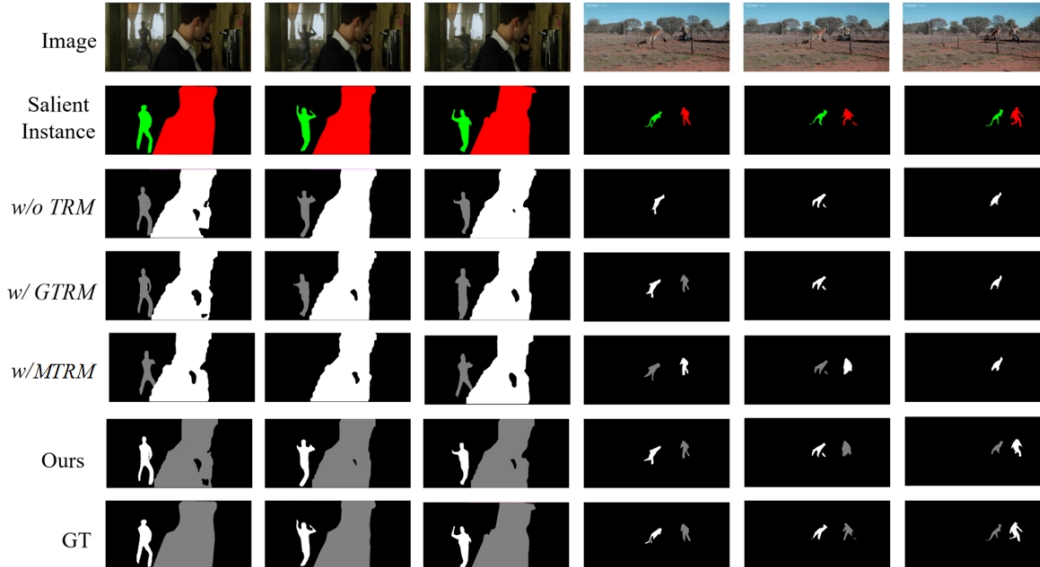

Figure 4: Visual results on ablation study. GTRM means global-aware temporal relation modeling and ITRM means instance-aware temporal relation modeling. It is obvious that our motion-aware features can effectively model the motion information of instances. The scenarios are chosen from $RVSOD/actioncliptrain00806$ and $DAVSOD/select\_0674$ respectively.

global features of the preceding and subsequent frames as temporal saliency cues. ITRM and MTRM represent instance interaction and motion-wise contrast in the method section respectively. Table 2 illustrates the large improvement by adding the motion-aware temporal interaction. It's obvious that either method greatly improves the ranking performance of the model compared to the baseline. Moreover, our proposed motion-aware feature is significantly superior to traditional temporal modules. Therefore, our final method combines the above three temporal features simultaneously.

In Figure 4, we observe that, when performing graph reasoning based purely on multi-scale spatial contrasts, the ranking shown in the $3^{th}$ row tends to favor static saliency cues (e.g., larger-sized individuals or those in closer proximity). Despite introducing global inter-frame contrast ("w/GTRM") and inter-frame cross-instance contrast ("w/ITRM") can capture some temporal cues, they are still unable to effectively model instance motion. However, by incorporating our motion-aware temporal correlation modeling into the graph, our method can effectively identify instances with noticeable motion cues, leading to successfully highlight of motion instance and superior performance.

Table 3: Quantitative comparison by varying the bounding box sizes.

| Expand Scale | $SA - SOR$ | |
| --- | --- | --- |
| | RVSOD | DAVSOD |
| $1\times$ | 0.6092 | 0.2421 |
| $2\times$ | **0.629** | **0.246** |
| $3\times$ | 0.5900 | 0.2374 |
| $4\times$ | 0.5684 | 0.2393 |

**Effectiveness on the local context size in motion-aware temporal relation modeling.** Without access to ground truth tracking annotations, it becomes quite challenging to model the motion trajectories of each individual instance accurately. Therefore, a motion-aware temporal relation modeling are proposed to evaluate the temporal saliency of the instance. However, as the instance moves and the camera shifts, simply projecting the bounding box onto adjacent frames is not enough to fully capture the changes in the instance. As shown in Table 3, double sized bounding box achieved the best results, while the excessive expand scale can easily introduce excessive background noise and degrade the results.

Table 4: ablation study for our fusion methods. Our baseline is the Liu's [2] model without person prior. Then we merge spatio-temporal relationships in three different orders.

| Method | SA-SOR | |
| --- | --- | --- |
| | RVSOD | DAVSOD |
| Baseline( w/o TRM ) | 0.563 | 0.179 |
| Spatial-then-temporal | 0.571 | 0.184 |
| Temporal-then-spatial | 0.590 | 0.183 |
| Ours(joint) | **0.603** | **0.207** |

**Effectiveness on the unified spatial-temporal reasoning.** Another question we want to investigate is how to make joint decisions based on spatial and temporal saliency cues. An intuitive choice is to forming the spatial and temporal graph separately and combine two graphs in a stage-wise manner. Hence, we involve two fusion variants for comparison. a) **Spatial-then-temporal**: start by aggregating three consecutive frames' spatial correlation graph independently and then fusion the temporal information using temporal correlation graph. b) **Temporal-then-spatial**: start by aggregating temporal correlation graph through the temporal context and then aggregate the spatial correlation graph with spatial context. c) **Our joint spatial-temporal graph reasoning**: It takes all features as inputs simultaneously and model their correlations adaptively for reasoning.

As shown in Table 4, two stage-wise reasoning strategies achieves similar improvement compared to the baseline that merely considers spatial-based reasoning, while our joint spatial-temporal reasoning strategy achieves a much higher improvement than them, demonstrating our method's better adaptivity and sufficiency in combining spatial and temporal saliency cues.

## 4.4   Application to Video Retargeting

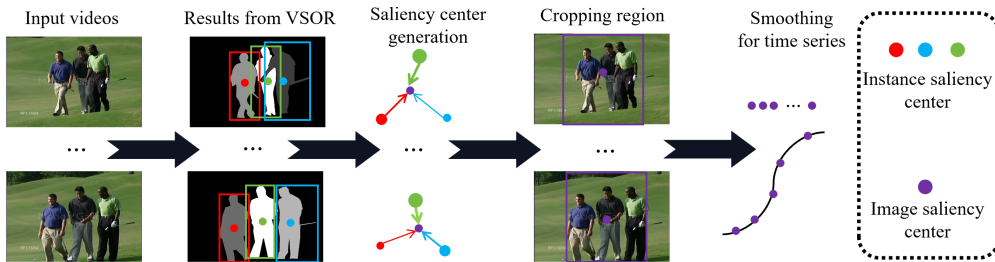

Figure 5: Our method for video retargeting. We generate instance-level saliency information including bounding box, mask and ranks based on our VSOR model. Thus, we get the cropping center according to instance-level saliency information.

With the development of multi-media technology, people can watch videos on different devices such as smartphones and tablets. Video retargeting adjusts the videos to different aspect ratios to suit different endpoints. The purpose of image retargeting is to preserve visually salient areas for a better visual experience. Based on this, image retargeting has been extremely successful in the past through methods such as seam carving [28–31], cropping [32–35] and warping [36–39]. However, simply applying these methods to video retargeting will lead to severe temporal inconsistency that greatly affect the human visual experience. Some works [12, 13] have been proposed to alleviate temporal inconsistency. For example, Zhu [12] utilize a dynamic spatial-temporal buffer to reduce temporal inconsistency, while Apostolidis [13] utilizes cropping for video retargeting based on saliency detection [40]. However, their method only distinguishes between foreground and background, ignoring the non-uniform saliency across instances in the foreground, thus usually resulting in biased retargeting to backgrounds or objects with less saliecncy.

Based on our VSOR, we can incorporate the saliency of instances into retargeting, thereby more adaptively preserving important semantic concepts in multi-object scenes. As shown in Figure 5, based on the rank of each salient instance with our VSOR model, We assign weights to each instance to get the cropping center. The center of instance i with $rank_i$ (the higher number means the more salient the instance i) is represented as $(x_i, y_i)$. Next based on the ranks of instances, we generate the

cropping center $(x_c, y_c)$ with the equation:

$$(x_c, y_c) = \left(\sum_{i=1}^{N} rank_i \times (x_i, y_i)\right) \Big/ \left(\sum_{i=1}^{N} rank_i\right) \tag{7}$$

Then we follow the smartVidCrop [13] to generate the crop windows for each frame and utilize the LOESS [41] to overcome visual mutations caused by camera shake.

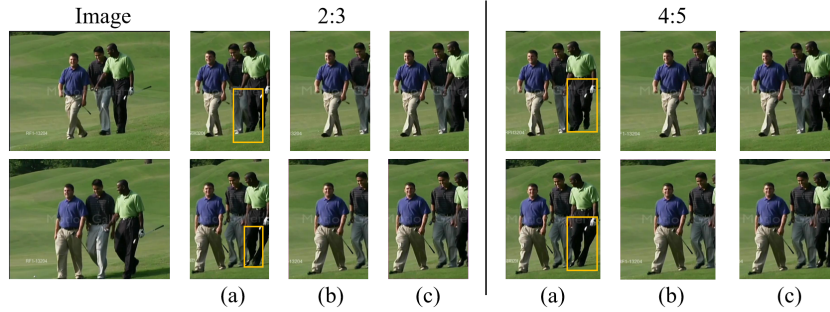

Figure 6: Comparison of retargeting, (a) is the method of seam carving [12], (b) is smartVideoCrop [13], (c) is our methods. The yellow box in (a) shows the artifacts after retargeting.

Our retargeting results are shown in Figure 6, we retarget the videos with the aspect ratio as 2:3 and 4:5 respectively. We compare our method with seam-carving based method proposed by Zhu [12] and crop-based method [13]. It is evident that our approach offers superior visual experience compared to the other methods. For instance, while the method in [12] detects all salient instances, it causes significant distortion to those instances, which negatively impacts the video quality. In contrast, Apostolidis's method [13] sometimes fails to crop out all the salient instances or gets distracted by the background information, resulting in inaccurate cropping windows. This indicates that our VSOR provides more precise semantic importance for the frames, allowing for more adaptive and reasonable results. Additional examples are available in the supplementary material in section A.3.

## 5   Conclusion

In this work, we propose a novel unified spatial-temporal graph that explicitly integrates trajectory-wise temporal correlation and adaptively integrate spatial and temporal saliency cues. Specifically, the first step involves extracting the features of each instance through an object detector and a feature enhancement module. And then a spatial-temporal graph reasoning network to fuse spatial-temporal saliency cues, ultimately obtaining saliency scores and final results. Finally, we applied our VSOR model to video retargeting and achieved impressive results.

**Limitations.** In our model, a detector is used to extract instance features. However, in different scenarios, the same object may exhibit different levels of saliency. For example, we should detect a tree in a desert but not in a forest. For non-salient objects, although our model assigns them a lower saliency ranking, we ideally do not want them to be detected at the first. In the future, we aim to design a saliency-sensitive detector to further improve performance.

## 6   Acknowledgement

This work was supported in part by the National Natural Science Foundation of China (NSFC) under Grant 62102083, Natural Science Foundation of Jiangsu Province under Grant BK20210222, the National Natural Science Foundation of China (NSFC) under Grant 62261160576, 62203024 and 92167102, and the R&D Program of Beijing Municipal Education Commission (KM202310005027).

# References

[1] Jiaying Lin, Huankang Guan, and Lau RynsonWH. Rethinking video salient object ranking. *arXiv preprint arXiv:2203.17257*, 2022.

[2] Nian Liu, Long Li, Wangbo Zhao, Junwei Han, and Ling Shao. Instance-level relative saliency ranking with graph reasoning. *IEEE Transactions on Pattern Analysis and Machine Intelligence*, 44(11):8321–8337, 2021.

[3] Md Amirul Islam, Mahmoud Kalash, and Neil DB Bruce. Revisiting salient object detection: Simultaneous detection, ranking, and subitizing of multiple salient objects. In *Proceedings of the IEEE Conference on Computer Vision and Pattern Recognition*, pages 7142–7150, 2018.

[4] Avishek Siris, Jianbo Jiao, Gary KL Tam, Xianghua Xie, and Rynson WH Lau. Inferring attention shift ranks of objects for image saliency. In *Proceedings of the IEEE/CVF Conference on Computer Vision and Pattern Recognition*, pages 12133–12143, 2020.

[5] Hao Fang, Daoxin Zhang, Yi Zhang, Minghao Chen, Jiawei Li, Yao Hu, Deng Cai, and Xiaofei He. Salient object ranking with position-preserved attention. In *Proceedings of the IEEE/CVF International Conference on Computer Vision*, pages 16331–16341, 2021.

[6] Xin Tian, Ke Xu, Xin Yang, Lin Du, Baocai Yin, and Rynson WH Lau. Bi-directional object-context prioritization learning for saliency ranking. In *Proceedings of the IEEE/CVF Conference on Computer Vision and Pattern Recognition*, pages 5882–5891, 2022.

[7] Nicole C Rust and Marlene R Cohen. Priority coding in the visual system. *Nature Reviews Neuroscience*, 23(6):376–388, 2022.

[8] Kelvin Xu, Jimmy Ba, Ryan Kiros, Kyunghyun Cho, Aaron Courville, Ruslan Salakhudinov, Rich Zemel, and Yoshua Bengio. Show, attend and tell: Neural image caption generation with visual attention. In *International Conference on Machine Learning*, pages 2048–2057. PMLR, 2015.

[9] Songtao Ding, Shiru Qu, Yuling Xi, and Shaohua Wan. Stimulus-driven and concept-driven analysis for image caption generation. *Neurocomputing*, 398:520–530, 2020.

[10] Li-Heng Chen, Christos G Bampis, Zhi Li, Andrey Norkin, and Alan C Bovik. Proxiqa: A proxy approach to perceptual optimization of learned image compression. *IEEE Transactions on Image Processing*, 30:360–373, 2020.

[11] David Tellez, Geert Litjens, Jeroen Van der Laak, and Francesco Ciompi. Neural image compression for gigapixel histopathology image analysis. *IEEE Transactions on Pattern Analysis and Machine Intelligence*, 43(2):567–578, 2019.

[12] Zhu Chuning. Fast video retargeting based on seam carving with parental labeling. *arXiv preprint arXiv:1903.03180*, 2019.

[13] Konstantinos Apostolidis and Vasileios Mezaris. A fast smart-cropping method and dataset for video retargeting. In *2021 IEEE International Conference on Image Processing (ICIP)*, pages 2618–2622. IEEE, 2021.

[14] Zheng Wang, Xinyu Yan, Yahong Han, and Meijun Sun. Ranking video salient object detection. In *Proceedings of the 27th ACM International Conference on Multimedia*, pages 873–881, 2019.

[15] Shaoqing Ren, Kaiming He, Ross Girshick, and Jian Sun. Faster r-cnn: Towards real-time object detection with region proposal networks. *Advances in Neural Information Processing Systems*, 28, 2015.

[16] Sung In Cho and Suk-Ju Kang. Temporal incoherence-free video retargeting using foreground aware extrapolation. *IEEE Transactions on Image Processing*, 29:4848–4861, 2020.

[17] Hyunwoo Nam, Dubok Park, and Kangwon Jeon. Jitter-robust video retargeting with kalman filter and attention saliency fusion network. In *2020 IEEE International Conference on Image Processing (ICIP)*, pages 858–862. IEEE, 2020.

[18] Christel Chamaret and Olivier Le Meur. Attention-based video reframing: Validation using eye-tracking. In *2008 19th International Conference on Pattern Recognition*, pages 1–4. IEEE, 2008.

[19] Haofeng Li, Guanqi Chen, Guanbin Li, and Yizhou Yu. Motion guided attention for video salient object detection. In *Proceedings of the IEEE/CVF International Conference on Computer Vision*, pages 7274–7283, 2019.

[20] Deng-Ping Fan, Wenguan Wang, Ming-Ming Cheng, and Jianbing Shen. Shifting more attention to video salient object detection. In *Proceedings of the IEEE/CVF Conference on Computer Vision and Pattern Recognition*, pages 8554–8564, 2019.

[21] Nian Liu, Kepan Nan, Wangbo Zhao, Xiwen Yao, and Junwei Han. Learning complementary spatial–temporal transformer for video salient object detection. *IEEE Transactions on Neural Networks and Learning Systems*, 2023.

[22] Kaiming He, Georgia Gkioxari, Piotr Dollár, and Ross Girshick. Mask r-cnn. In *Proceedings of the IEEE International Conference on Computer Vision*, pages 2961–2969, 2017.

[23] Sanghyun Woo, Jongchan Park, Joon-Young Lee, and In So Kweon. Cbam: Convolutional block attention module. In *Proceedings of the European Conference on Computer Vision*, pages 3–19, 2018.

[24] Chengxiao Sun, Yan Xu, Jialun Pei, Haopeng Fang, and He Tang. Partitioned saliency ranking with dense pyramid transformers. In *Proceedings of the 31st ACM International Conference on Multimedia*, pages 1874–1883, 2023.

[25] Huankang Guan and Rynson WH Lau. Seqrank: Sequential ranking of salient objects. In *Proceedings of the AAAI Conference on Artificial Intelligence*, volume 38, pages 1941–1949, 2024.

[26] Miao Zhang, Jie Liu, Yifei Wang, Yongri Piao, Shunyu Yao, Wei Ji, Jingjing Li, Huchuan Lu, and Zhongxuan Luo. Dynamic context-sensitive filtering network for video salient object detection. In *Proceedings of the IEEE/CVF international conference on computer vision*, pages 1553–1563, 2021.

[27] Lihao Liu, Jean Prost, Lei Zhu, Nicolas Papadakis, Pietro Liò, Carola-Bibiane Schönlieb, and Angelica I Aviles-Rivero. Scotch and soda: A transformer video shadow detection framework. In *Proceedings of the IEEE/CVF Conference on Computer Vision and Pattern Recognition*, pages 10449–10458, 2023.

[28] Yanfei Peng, Jing Wang, Xiaoxuan Liu, and Shengjie Gong. Seam carving algorithm based on partitioning. In *2023 IEEE International Conference on Control, Electronics and Computer Technology*, pages 1173–1176. IEEE, 2023.

[29] Srishti Sharma and Yatharth Piplani. Comparative analysis of seam carving in images. In *Proceedings of the Second International Conference on Information Management and Machine Intelligence*, pages 139–146. Springer, 2021.

[30] Shai Avidan and Ariel Shamir. Seam carving for content-aware image resizing. In *Seminal Graphics Papers: Pushing the Boundaries, Volume 2*, pages 609–617. 2023.

[31] Mahdi Hashemzadeh, Bahareh Asheghi, and Nacer Farajzadeh. Content-aware image resizing: An improved and shadow-preserving seam carving method. *Signal Processing*, 155:233–246, 2019.

[32] Li-Qun Chen, Xing Xie, Xin Fan, Wei-Ying Ma, Hong-Jiang Zhang, and He-Qin Zhou. A visual attention model for adapting images on small displays. *Multimedia Systems*, 9:353–364, 2003.

[33] Hao Liu, Xing Xie, Wei-Ying Ma, and Hong-Jiang Zhang. Automatic browsing of large pictures on mobile devices. In *Proceedings of the eleventh ACM International Conference on Multimedia*, pages 148–155, 2003.

[34] Anthony Santella, Maneesh Agrawala, Doug DeCarlo, David Salesin, and Michael Cohen. Gaze-based interaction for semi-automatic photo cropping. In *Proceedings of the SIGCHI Conference on Human Factors in Computing Systems*, pages 771–780, 2006.

[35] Bongwon Suh, Haibin Ling, Benjamin B Bederson, and David W Jacobs. Automatic thumbnail cropping and its effectiveness. In *Proceedings of the 16th annual ACM Symposium on User Interface Software and Technology*, pages 95–104, 2003.

[36] Yu-Shuen Wang, Chiew-Lan Tai, Olga Sorkine, and Tong-Yee Lee. Optimized scale-and-stretch for image resizing. In *ACM SIGGRAPH Asia 2008 papers*, pages 1–8. 2008.

[37] Yanwen Guo, Feng Liu, Jian Shi, Zhi-Hua Zhou, and Michael Gleicher. Image retargeting using mesh parametrization. *IEEE Transactions on Multimedia*, 11(5):856–867, 2009.

[38] Bing Li, Yiming Chen, Jinqiao Wang, Ling-Yu Duan, and Wen Gao. Fast retargeting with adaptive grid optimization. In *2011 IEEE International Conference on Multimedia and Expo*, pages 1–4. IEEE, 2011.

[39] Pierre-Yves Laffont, Jong Yun Jun, Christian Wolf, Yu-Wing Tai, Khalid Idrissi, George Drettakis, and Sung-Eui Yoon. Interactive content-aware zooming. In *Graphics Interface 2010 Conference*, pages 79–87. Canadian Information Processing Society Toronto, Ont., Canada, 2010.

[40] Richard Droste, Jianbo Jiao, and J Alison Noble. Unified image and video saliency modeling. In *Computer Vision–ECCV 2020: 16th European Conference, Glasgow, UK, August 23–28, 2020, Proceedings, Part V 16*, pages 419–435. Springer, 2020.

[41] William S Cleveland and Susan J Devlin. Locally weighted regression: an approach to regression analysis by local fitting. *Journal of the American Statistical Association*, 83(403):596–610, 1988.

# A Supplemental Material

## A.1 Details of Dataset Generation

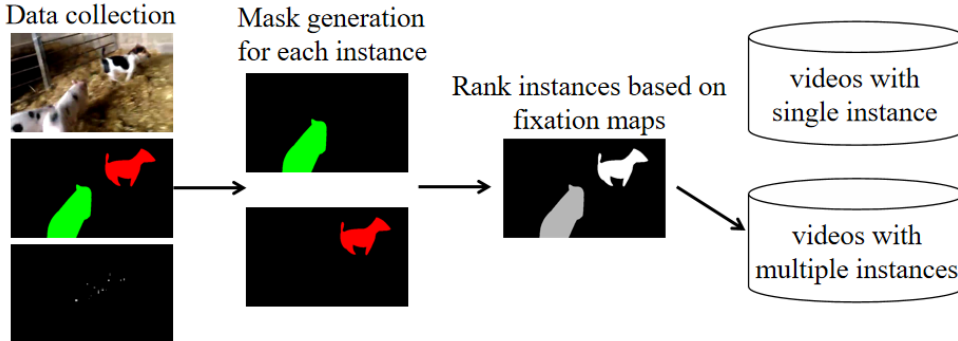

Figure 7: Flowchat for dataset collection and annotation.

Table 5: Statistical analysis of DAVSOD dataset for video salient object ranking. Semi valid indicates that some video frames in the scene only have one salient instance, valid indicates that the scene or video frame contains multiple salient instances, and invalid indicates that the video frame only contains one salient instance.

| Dataset | Number of scenes | | Number of images | |
|---|---|---|---|---|
| | semi valid | valid | invalid | valid |
| Training set | 53 | 40 | 2469 | 9067 |
| Test set | 13 | 9 | 723 | 2364 |

As shown in Figure 7, We select videos with varying object numbers and then determine the saliency rank by the given instance masks and comparing the given fixation numbers for each instance. Table 5 represents the statistical analysis of DAVSOD dataset for video salient object ranking. Semi valid indicates that some video frames in the scene only have one salient instance, valid indicates that the scene or video frame contains multiple salient instances, and invalid indicates that the video frame only contains one salient instance.

## A.2 Analysis for Poor Results on DAVSOD

Table 6: Analysis for DAVSOD test set.

| Category | SA-SOR | mAP | Proportion | Example |
|---|---|---|---|---|
| (a) hard to detect | -0.07 | 0.50 | 5/22 | select_0557, select_0208, select_0572 |
| (b) low quality of labeling | 0.16 | 0.44 | 7/22 | select_0607, select_0345, select_0577 |
| (c) others | 0.45 | 0.70 | 10/22 | |

We perform an in-depth study of the DAVSOD test set in Table 6. The results in the table below classify scenes by challenge, proportion, and examples, and report ranking (SA-SOR) and detection (mAP) performance using our model trained on the full DAVSOD training set. Analysis of these results reveals two key reasons for the low performance:

a) Severe occlusion among multiple objects or objects with very small sizes, making it difficult to successfully detect all salient objects (e.g., the instructor and the person skydiving with blocked each other are perceived as a single entity in the $select\_0557$ video). Similar data includes: $select\_0208$, $select\_0557$, $select\_0590\_3$, $select\_0257$ and $select\_0572$.

b) Severe variance in salient objects between adjacent frames: The DAVSOD salient object annotations are based on subjective eye fixations from multiple testers, which exhibit increased variance as the number of objects increases. This results in inconsistent and unreliable SOD and ranking labels, as fixations shift significantly across frames in scenes with diverse objects. Compared to RVSOD,

DAVSOD contains much more such scenes with multiple objects, where the saliency of individual objects can flicker between salient and non-salient in adjacent frames (e.g., the bull and person in the *select*_0607 bullfighting video). Similar data includes: *select*_0607, *select*_0001, *select*_0297, *select*_0653, *select*_0577, *select*_0345 and *select*_0674.

### A.3 More Visual Results of Video Retargeting

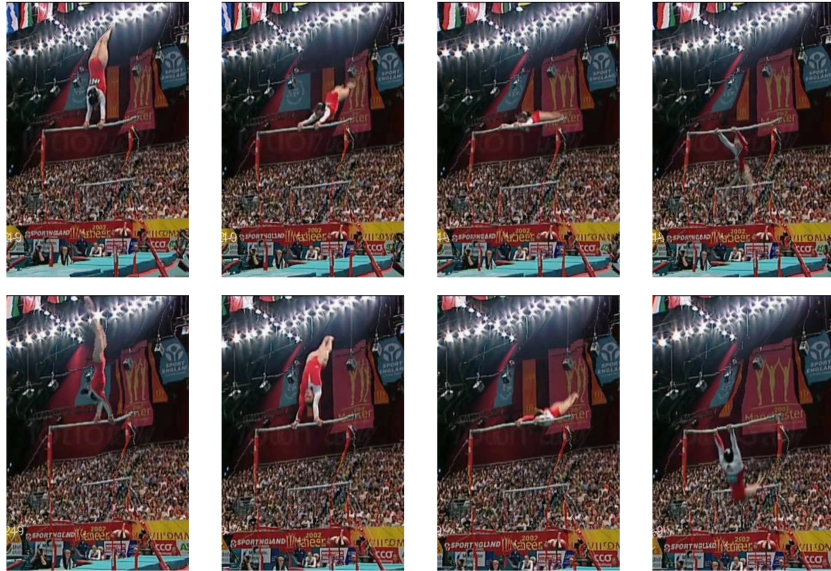

Figure 8: The results of the seam-carving based method [12].

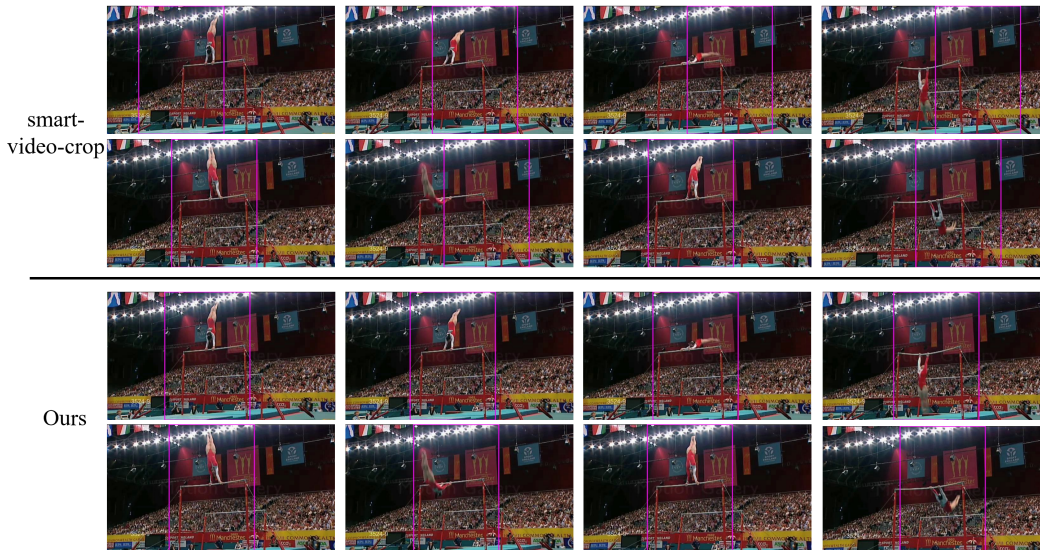

Figure 9: The results of the cropping based methods. The upper part is the smart-video-crop [13], and the lower part is our model. The red boxes in the images represent the crop area.

As shown in Figure 8, the seam carving based method [12] can easily cause image distortion which will result in extremely poor visual experiences. While in Figure 9, cropping based methods solve this problem. However, previous work [13] utilized image-level saliency for cropping, making the model is sensitive to noise and resulting in inaccurate cropping of salient objects. On the contrary, our method adopts instance-level saliency, keeping the salient object at the center of the cropping box, thereby greatly improving the human visual experience.

